# The Recurrent Temporal Restricted Boltzmann Machine

**Ilya Sutskever, Geoffrey Hinton, and Graham Taylor**
University of Toronto
{ilya, hinton, gwtaylor}@cs.utoronto.ca

## Abstract

The Temporal Restricted Boltzmann Machine (TRBM) is a probabilistic model for sequences that is able to successfully model (i.e., generate nice-looking samples of) several very high dimensional sequences, such as motion capture data and the pixels of low resolution videos of balls bouncing in a box. The major disadvantage of the TRBM is that exact inference is extremely hard, since even computing a Gibbs update for a single variable of the posterior is exponentially expensive. This difficulty has necessitated the use of a heuristic inference procedure, that nonetheless was accurate enough for successful learning. In this paper we introduce the Recurrent TRBM, which is a very slight modification of the TRBM for which exact inference is very easy and exact gradient learning is almost tractable. We demonstrate that the RTRBM is better than an analogous TRBM at generating motion capture and videos of bouncing balls.

## 1 Introduction

Modeling sequences is an important problem since there is a vast amount of natural data, such as speech and videos, that is inherently sequential. A good model for these data sources could be useful for finding an abstract representation that is helpful for solving "natural" discrimination tasks (see [4] for an example of this approach for the non-sequential case). In addition, it could be also used for predicting the future of a sequence from its past, be used as a prior for denoising tasks, and be used for other applications such as tracking objects in video. The Temporal Restricted Boltzmann Machine [14, 13] is a recently introduced probabilistic model that has the ability to accurately model complex probability distributions over high-dimensional sequences. It was shown to be able to generate realistic motion capture data [14], and low resolution videos of 2 balls bouncing in a box [13], as well as complete and denoise such sequences.

As a probabilistic model, the TRBM is a directed graphical model consisting of a sequence of Restricted Boltzmann Machines (RBMs) [3], where the state of one or more previous RBMs determines the biases of the RBM in next timestep. This probabilistic formulation straightforwardly implies a learning procedure where approximate inference is followed by learning. The learning consists of learning a conditional RBM at each timestep, which is easily done with Contrastive Divergence (CD) [3]. Exact inference in TRBMs, on the other hand, is highly non-trivial, since computing even a single Gibbs update requires computing the ratio of two RBM partition functions. The approximate inference procedure used in [13] was heuristic and was not even derived from a variational principle.

In this paper we introduce the Recurrent TRBM (RTRBM), which is a model that is very similar to the TRBM, and just as expressive. Despite the similarity, *exact* inference is very easy in the RTRBM and computing the gradient of the log likelihood is feasible (up to the error introduced by the use of Contrastive Divergence). We demonstrate that the RTRBM is able to generate more realistic samples than an equivalent TRBM for the motion capture data and for the pixels of videos

of bouncing balls. The RTRBM's performance is better than the TRBM mainly because it learns to convey more information through its hidden-to-hidden connections.

## 2   Restricted Boltzmann Machines

The building block of the TRBM and the RTRBM is the Restricted Boltzmann Machine [3]. An RBM defines a probability distribution over pairs of vectors, $V \in \{0,1\}^{N_V}$ and $H \in \{0,1\}^{N_H}$ (a shorthand for visible and hidden) by the equation

$$P(v,h) = P(V=v, H=h) = \exp(v^\top b_V + h^\top b_H + v^\top W h)/Z \qquad (1)$$

where $b_V$ is a vector of biases for the visible vectors, $b_H$ is a vector of biases for the hidden vectors, and $W$ is the matrix of connection weights. The quantity $Z = Z(b_V, b_H, W)$ is the value of the partition function that ensures that Eq. 1 is a valid probability distribution. The RBM's definition implies that the conditional distributions $P(H|v)$ and $P(V|h)$ are factorial (i.e., all the components of $H$ in $P(H|v)$ are independent) and are given by $P(H^{(j)} = 1|v) = s(b_H + W^\top v)^{(j)}$ and $P(V^{(i)} = 1|h) = s(b_V + Wh)^{(i)}$, where $s(x)^{(j)} = (1 + \exp(-x^{(j)}))^{-1}$ is the logistic function and $x^{(j)}$ is the $j$th component of the vector $x$. In general, we use $i$ to index visible vectors $V$ and $j$ to index hidden vectors $H$. [1] The RBM can be slightly modified to allow the vector $V$ to take real values; one way of achieving this is by the definition

$$P(v,h) = \exp(-\|v\|^2/2 + v^\top b_V + h^\top b_H + v^\top W h)/Z. \qquad (2)$$

Using this equation does not change the form of the gradients and the conditional distribution $P(H|v)$. The only change it introduces is in the conditional distribution $P(V|h)$, which is equal to a multivariate Gaussian with parameters $\mathcal{N}(b_V + Wh, \mathbf{I})$. See [18, 14] for more details and generalizations.

The gradient of the average log probability given a dataset $S$, $L = 1/|S| \sum_{v \in S} \log P(v)$, has the following simple form:

$$\partial L/\partial W \quad = \quad \left\langle V \cdot H^\top \right\rangle_{P(H|V)\tilde{P}(V)} - \left\langle V \cdot H^\top \right\rangle_{P(H,V)} \qquad (3)$$

where $\tilde{P}(V) = 1/|S| \sum_{v \in S} \delta_v(V)$ (here $\delta_x(X)$ is a distribution over real-valued vectors that is concentrated at $x$), and $\langle f(X) \rangle_{P(X)}$ is the expectation of $f(X)$ under the distribution $P$. Computing the exact values of the expectations $\langle \cdot \rangle_{P(H,V)}$ is computationally intractable, and much work has been done on methods for computing approximate values for the expectations that are good enough for practical learning and inference tasks (e.g., [16, 12, 19], including [15], which works well for the RBM).

We will approximate the gradients with respect to the RBM's parameters using the Contrastive Divergence [3] learning procedure, $CD_n$, whose updates are computed by the following algorithm.

**Algorithm 1 ($CD_n$)**

    1. Sample $(v,h) \sim P(H|V)\tilde{P}(V)$
    2. Set $\Delta W$ to $v \cdot h^\top$
    3. **repeat** $n$ times: sample $v \sim P(V|h)$, then sample $h \sim P(H|v)$
    4. Decrease $\Delta W$ by $v \cdot h^\top$

Models learned by $CD_1$ are often reasonable generative models of the data [3], but if learning is continued with $CD_{25}$, the resulting generative models are much better [11]. The RBM also plays a critical role in deep belief networks [4], [5], but we do not use this connection in this paper.

## 3   The TRBM

It is easy to construct the TRBM with RBMs. The TRBM, as described in the introduction, is a sequence of RBMs arranged in such a way that in any given timestep, the RBM's biases depend only on the state of the RBM in the previous timestep. In its simplest form, the TRBM can

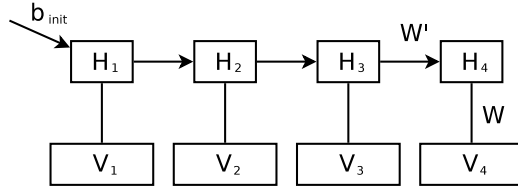

Figure 1: The graphical structure of a TRBM: a directed sequence of RBMs.

be viewed as a Hidden Markov Model (HMM) [9] with an exponentially large state space that has an extremely compact parameterization of the transition and the emission probabilities. Let $X_{t_A}^{t_B} = (X_{t_A}, \dots, X_{t_B})$ denote a sequence of variables. The TRBM defines a probability distribution $P(V_1^T = v_1^T, H_1^T = h_1^T)$ by the equation

$$P(v_1^T, h_1^T) = \prod_{t=2}^{T} P(v_t, h_t | h_{t-1}) P_0(v_1, h_1) \tag{4}$$

which is identical to the defining equation of the HMM. The conditional distribution $P(V_t, H_t | h_{t-1})$ is that of an RBM, whose biases for $H_t$ are a function of $h_{t-1}$. Specifically,

$$P(v_t, h_t | h_{t-1}) = \exp\left(v_t^\top b_V + v_t^\top W h_t + h_t^\top (b_H + W' h_{t-1})\right) / Z(h_{t-1}) \tag{5}$$

where $b_V$, $b_H$ and $W$ are as in Eq. 1, while $W'$ is the weight matrix of the connections from $H_{t-1}$ to $H_t$, making $b_H + W' h_{t-1}$ be the bias of RBM at time $t$. In this equation, $V \in \{0,1\}^{N_V}$ and $H \in \{0,1\}^{N_H}$; it is easy to modify this definition to allow $V$ to take real values as was done in Eq. 2. The RBM's partition function depends on $h_{t-1}$, because the parameters (i.e., the biases) of the RBM at time $t$ depend on the value of the random variable $H_{t-1}$. Finally, the distribution $P_0$ is defined by an equation very similar to Eq. 5, except that the (undefined) term $W' h_0$ is replaced by the term $b_{init}$, so the hidden units receive a special initial bias at $P_0$; we will often write $P(V_1, H_1 | h_0)$ for $P_0(V_1, H_1)$ and $W' h_0$ for $b_{init}$. It follows from these equations that the TRBM is a directed graphical model that has an (undirected) RBM at each timestep (a related directed sequence of Boltzmann Machines has been considered in [7]).

As in most probabilistic models, the weight update is computed by solving the inference problem and computing the weight update as if the inferred variables were observed. fully-visible case. If the hidden variables are observed, equation 4 implies that the gradient of the log likelihood with respect to the TRBM's parameters is $\sum_{t=1}^{T} \nabla \log P(v_t, h_t | h_{t-1})$, and each term, being the gradient of the log likelihood of an RBM, can be approximated using $\text{CD}_n$. Thus the main computational difficulty of learning TRBMs is in obtaining samples from a distribution approximating the posterior $P(H_1^T | v_1^T)$.

**Inference in a TRBM**

Unfortunately, the TRBM's inference problem is harder than that of a typical undirected graphical model, because even computing the probability $P(H_t^{(j)} = 1 | \text{everything else})$ involves evaluating the exact ratio of two RBM partition functions, which can be seen from Eq. 5. This difficulty necessitated the use of a heuristic inference procedure [13], which is based on the observation that the distribution $P(H_t | h_1^{t-1}, v_1^t) = P(H_t | h_{t-1}, v_t)$ is factorial by definition. This inference procedure does not do any kind of smoothing from the future and only does approximate filtering from the past by sampling from the distribution $\prod_{t=1}^{T} P(H_t | H_1^{t-1}, v_1^t)$ instead of the true posterior distribution $\prod_{t=1}^{T} P(H_t | H_1^{t-1}, v_1^T)$, which is easy because $P(H_t | h_1^{t-1}, v_1^t)$ is factorial. [2]

## 4   Recurrent TRBMs

Let us start with notation. Consider an arbitrary factorial distribution $P'(H)$. The statement $h \sim P'(H)$ means that $h$ is sampled from the factorial distribution $P'(H)$, so each $h^{(j)}$ is set to 1 with

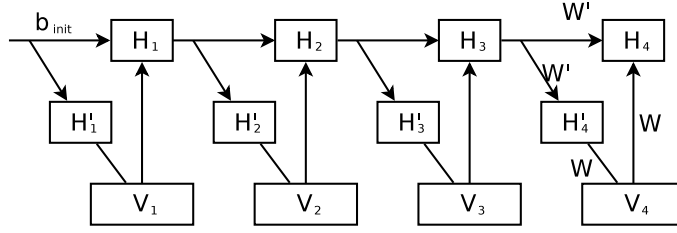

Figure 2: The graphical structure of the RTRBM, $Q$. The variables $H_t$ are real valued while the variables $H_t'$ are binary. The conditional distribution $Q(V_t, H_t'|h_{t-1})$ is given by the equation $Q(v_t, h_t'|h_{t-1}) = \exp\left(v_t^\top W h_t' + v_t^\top b_V + h_t'(b_H + W' h_{t-1})\right)/Z(h_{t-1})$, which is essentially the same as the TRBM's conditional distribution $P$ from equation 5. We will always integrate out $H_t'$ and will work directly with the distribution $Q(V_t|h_{t-1})$. Notice that when $V_1$ is observed, $H_1'$ cannot affect $H_1$.

probability $P'(H^{(j)} = 1)$, and 0 otherwise. In contrast, the statement $h \leftarrow P'(H)$ means that each $h^{(j)}$ is set to the real value $P'(H^{(j)} = 1)$, so this is a "mean-field" update [8, 17]. The symbol $P$ stands for the distribution of some TRBM, while the symbol $Q$ stands for the distribution defined by an RTRBM. Note that the outcome of the operation $\cdot \leftarrow P(H_t|v_t, h_{t-1})$ is $s(Wv_t + W'h_{t-1} + b_H)$.

An RTRBM, $Q(V_1^T, H_1^T)$, is defined by the equation

$$Q(v_1^T, h_1^T) = \prod_{t=2}^{T} Q(v_t|h_{t-1})Q(h_t|v_t, h_{t-1}) \cdot Q_0(v_1) \cdot Q_0(h_1|v_1) \tag{6}$$

The terms appearing in this equation will be defined shortly.

Let us contrast the generative process of the two models. To sample from a TRBM $P$, we need to perform a directed pass, sampling from each RBM on every timestep. One way of doing this is described by the following algorithm.

**Algorithm 2 (for sampling from the TRBM):**

**for** $1 \leq t \leq T$:

      1. sample $v_t \sim P(V_t|h_{t-1})$
      2. sample $h_t \sim P(H_t|v_t, h_{t-1})$ [3]

where step 1 requires sampling from the marginals of a Boltzmann Machine (by integrating out $H_t$), which involves running a Markov chain.

By definition, RTRBMs and TRBMs are parameterized in the same way, so from now on we will assume that $P$ and $Q$ have identical parameters, which are $W, W', b_V, b_H$, and $b_{init}$. The following algorithm samples from the RTRBM $Q$ under this assumption.

**Algorithm 3 (for sampling from the RTRBM)**

**for** $1 \leq t \leq T$:

      1. sample $v_t \sim P(V_t|h_{t-1})$
      2. set $h_t \leftarrow P(H_t|v_t, h_{t-1})$

We can infer that $Q(V_t|h_{t-1}) = P(V_t|h_{t-1})$ because of step 1 in Algorithm 3, which is also consistent with the equation given in figure 2 where $H_t'$ is integrated out. The only difference between Algorithm 2 and Algorithm 3 is in step 2. The difference may seem small, since the operations $h_t \sim P(H_t|v_t, h_{t-1})$ and $h_t \leftarrow P(H_t|v_t, h_{t-1})$ appear similar. However, this difference significantly alters the inference and learning procedures of the RTRBM; in particular, it can already be seen that $H_t$ are real-valued for the RTRBM.

## 4.1 Inference in RTRBMs

Inference in RTRBMs given $v_1^T$ is very easy, which might be surprising in light of its similarity to the TRBM. The reason inference is easy is similar to the reason inference in square ICAs is easy [1]: There is a *unique* and an *easily computable* value of the hidden variables that has a nonzero posterior probability. Suppose, for example, that the value of $V_1$ is $v_1$, which means that $v_1$ was produced at the end of step 1 in Algorithm 3. Since step 2, the deterministic operation $h_1 \leftarrow P_0(H_1|v_1)$, has been executed, the only value $h_1$ can take is the value assigned by the operation $\cdot \leftarrow P_0(H_1|v_1)$. Any other value for $h_1$ is never produced by a generative process that outputs $v_1$ and thus has posterior probability 0. In addition, by executing this operation, we can recover $h_1$. Thus, $Q_0(H_1|v_1) = \delta_{s(Wv_1+b_H+b_{init})}(H_1)$. Note that $H_1$'s value is completely independent of $v_2^T$.

Once $h_1$ is known, we can consider the generative process that produced $v_2$. As before, since $v_2$ was produced at the end of step 1, then the fact that step 2 has been executed implies that $h_2$ can be computed by $h_2 \leftarrow P(H_2|v_2, h_1)$ (recall that at this point $h_1$ is known with absolute certainty). If the same reasoning is repeated $t$ times, then all of $h_1^t$ is uniquely determined and is easily computed when $V_1^t$ is known. There is no need for smoothing because $V_t$ and $H_{t-1}$ influence $H_t$ with such strength that the knowledge of $V_{t+1}^T$ cannot alter the model's belief about $H_t$. This is because $Q(H_t|v_t, h_{t-1}) = \delta_{s(Wv_t+b_H+W'h_{t-1})}(H_t)$.

The resulting inference algorithm is simple:

**Algorithm 4 (inference in RTRBMs)**

**for** $1 \le t \le T$:

      1. $h_t \leftarrow P(H_t|v_t, h_{t-1})$

Let $h(v)_1^T$ denote the output of the inference algorithm on input $v_1^T$, in which case the posterior is described by

$$Q(H_1^T|v_1^T) = \delta_{h(v)_1^T}(H_1^T). \tag{7}$$

## 4.2 Learning in RTRBMs

Learning in RTRBMs may seem easy once inference is solved, since the main difficulty in learning TRBMs is the inference problem. However, the RTRBM does not allow EM-like learning because the equation $\nabla \log Q(v_1^T) = \langle \nabla \log Q(v_1^T, h_1^T) \rangle_{h_1^T \sim Q(H_1^T|v_1^T)}$ is not meaningful. To be precise, the gradient $\nabla \log Q(v_1^T, h_1^T)$ is undefined because $\delta_{s(W'h_{t-1}+b_H+W^Tv_t)}(h_t)$ is not, in general, a continuous function of $W$. Thus, the gradient has to be computed differently.

Notice that the RTRBM's log probability satisfies $\log Q(v_1^T) = \sum_{t=1}^T \log Q(v_t|v_1^{t-1})$, so we could try computing the sum $\nabla \sum_{t=1}^T \log Q(v_t|v_1^{t-1})$. The key observation that makes the computation feasible is the equation

$$Q(V_t|v_1^{t-1}) = Q(V_t|h(v)_{t-1}) \tag{8}$$

where $h(v)_{t-1}$ is the value computed by the RTRBM inference algorithm with inputs $v_1^t$. This equation holds because $Q(v_t|v_1^{t-1}) = \int_{h'_{t-1}} Q(v_t|h'_{t-1})Q(h'_{t-1}|v_1^{t-1})dh'_{t-1} = Q(v_t|h(v)_{t-1})$, as the posterior distribution $Q(H_{t-1}|v_1^{t-1}) = \delta_{h(v)_{t-1}}(H_{t-1})$ is a point-mass at $h(v)_{t-1}$, which follows from Eq. 7.

The equality $Q(V_t|v_1^{t-1}) = Q(V_t|h(v)_{t-1})$ allows us to define a recurrent neural network (RNN) [10] whose parameters are identical to those of the RTRBM, and whose cost function is equal to the log likelihood of the RTRBM. This is useful because it is easy to compute gradients with respect to the RNN's parameters using the backpropagation through time algorithm [10]. The RNN has a pair of variables at each timestep, $\{(v_t, r_t)\}_{t=1}^T$, where $v_t$ are the input variables and $r_t$ are the RNN's hidden variables (all of which are deterministic). The hiddens $r_1^T$ are computed by the equation

$$r_t = s(Wv_t + b_H + W'r_{t-1}) \tag{9}$$

where $W'r_{t-1}$ is replaced with $b_{init}$ when $t = 1$. This definition was chosen so that the equation $r_1^T = h(v)_1^T$ would hold. The RNN attempts to probabilistically predict the next timestep from its history using the marginal distribution of the RBM $Q(V_{t+1}|r_t)$, so its objective function at time $t$ is defined to be $\log Q(v_{t+1}|r_t)$, where $Q$ depends on the RNN's parameters in the same way it depends

on the RTRBM's parameters (the two sets of parameters being identical). This is a valid definition of an RNN whose cumulative objective for the sequence $v_1^T$ is

$$O = \sum_{t=1}^{T} \log Q(v_t|r_{t-1}) \tag{10}$$

where $Q(v_1|r_0) = Q_0(v_1)$. But since $r_t$ as computed in equation 9 on input $v_1^T$ is identical to $h(v)_t$, the equality $\log Q(v_t|r_{t-1}) = \log Q(v_t|v_1^{t-1})$ holds. Substituting this identity into Eq. 10 yields

$$O = \sum_{t=1}^{T} \log Q(v_t|r_{t-1}) = \sum_{t=1}^{T} \log Q(v_t|v_1^{t-1}) = \log Q(v_1^T) \tag{11}$$

which is the log probability of the corresponding RTRBM.

This means that $\nabla O = \nabla \log Q(v_1^T)$ can be computed with the backpropagation through time algorithm [10], where the contribution of the gradient from each timestep is computed with Contrastive Divergence.

### 4.3   Details of the backpropagation through time algorithm

The backpropagation through time algorithm is identical to the usual backpropagation algorithm where the feedforward neural network is turned "on its side". Specifically, the algorithm maintains a term $\partial O/\partial r_t$ which is computed from $\partial O/\partial r_{t+1}$ and $\partial \log Q(v_{t+1}|r_t)/\partial r_t$ using the chain rule, by the equation

$$\partial O/\partial r_t = {W'}^\top (r_{t+1}.(1 - r_{t+1}).\partial O/\partial r_{t+1}) + {W'}^\top \partial \log Q(v_t|r_{t-1})/\partial b_H \tag{12}$$

where $a.b$ denotes component-wise multiplication, the term $r_t.(1 - r_t)$ arises from the derivative of the logistic function $s'(x) = s(x).(1 - s(x))$, and $\partial \log Q(v_{t+1}|r_t)/\partial b_H$ is computed by CD. Once $\partial O/\partial r_t$ is computed for all $t$, the gradients of the parameters can be computed using the following equations

$$\frac{\partial O}{\partial W'} = \sum_{t=2}^{T} r_{t-1}(r_t.(1 - r_t).\partial O/\partial r_t)^\top \tag{13}$$

$$\frac{\partial O}{\partial W} = \sum_{t=1}^{T-1} v_t \left( {W'}^\top (r_{t+1}.(1 - r_{t+1}).\partial O/\partial r_{t+1}) \right)^\top + \sum_{t=1}^{T} \partial \log Q(v_t|r_{t-1})/\partial W \tag{14}$$

The first summation in Eq. 14 arises from the use of $W$ as weights for inference for computing $r_t$ and the second summation arises from the use of $W$ as RBM parameters for computing $\log Q(v_t|r_{t-1})$. Each term of the form $\partial \log Q(v_{t+1}|r_t)/\partial W$ is also computed with CD. Computing $\partial O/\partial r_t$ is done most conveniently with a single backward pass through the sequence. As always, $\log Q(v_1|r_0) = Q_0(v_1)$. It is also seen that the gradient would be computed exactly if CD were to return the exact gradient of the RBM's log probability.

## 5   Experiments

We report the results of experiments comparing an RTRBM to a TRBM. The results in [14, 13] were obtained using TRBMs that had several delay-taps, which means that each hidden unit could directly observe several previous timesteps. To demonstrate that the RTRBM learns to use the hidden units to store information, we did not use delay-taps for the RTRBM nor the TRBM, which causes the results to be worse (but not much) than in [14, 13]. If delay-taps are allowed, then the results of [14, 13] show that there is little benefit from the hidden-to-hidden connections (which are $W'$), making the comparison between the RTRBM and the TRBM uninteresting.

In all experiments, the RTRBM and the TRBM had the same number of hidden units, their parameters were initialized in the same manner, and they were trained for the same number of weight updates. When sampling from the TRBM, we would use the sampling procedure of the RTRBM using the TRBM's parameters to eliminate the additional noise from its hidden units. If this is not done, the samples produced by the TRBM are significantly worse. Unfortunately, the evaluation metric is entirely qualitative since computing the log probability on a test set is infeasible for both the TRBM and the RTRBM. We provide the code for our experiments in [URL].

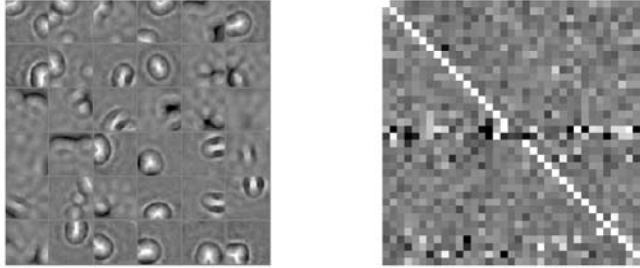

Figure 3: This figure shows the receptive fields of the first 36 hidden units of the RTRBM on the left, and the corresponding hidden-to-hidden weights between these units on the right: the $i$th row on the right corresponds to the $i$th receptive field on the left, when counted left-to-right. Hidden units 18 and 19 exhibit unusually strong hidden-to-hidden connections; they are also the ones with the weakest visible-hidden connections, which effectively makes them belong to another hidden layer.

## 5.1 Videos of bouncing balls

We used a dataset consisting of videos of 3 balls bouncing in a box. The videos are of length 100 and of resolution $30 \times 30$. Each training example is synthetically generated, so no training sequence is seen twice by the model which means that overfitting is highly unlikely. The task is to learn to generate videos at the pixel level. This problem is high-dimensional, having 900 dimensions per frame, and the RTRBM and the TRBM are given no prior knowledge about the nature of the task (e.g., by convolutional weight matrices).

Both the RTRBM and the TRBM had 400 hidden units. Samples from these models are provided as videos 1,2 (RTRBM) and videos 3,4 (TRBM). A sample training sequence is given in video 5. All the samples can be found in [URL]. The real-values in the videos are the conditional probabilities of the pixels [13]. The RTRBM's samples are noticeably better than the TRBM's samples; a key difference between these samples is that the balls produced by the TRBM moved in a random walk, while those produced by the RTRBM moved in a more persistent direction. An examination of the visible to hidden connection weights of the RTRBM reveals a number of hidden units that are not connected to visible units. These units have the most active hidden to hidden connections, which must be used to propagate information through time. In particular, these units are the only units that do not have a strong self connection (i.e., $W'_{i,i}$ is not large; see figure 3). No such separation of units is found in the TRBM and all its hidden units have large visible to hidden connections.

## 5.2 Motion capture data

We used a dataset that represents human motion capture data by sequences of joint angle, translations, and rotations of the base of the spine [14]. The total number of frames in the dataset set was 3000, from which the model learned on subsequences of length 50. Each frame has 49 dimensions, and both models have 200 hidden units. The data is real-valued, so the TRBM and the RTRBM were adapted to have Gaussian visible variables using equation 2. The samples produced by the RTRBM exhibit less sticking and foot-skate than those produced by the TRBM; samples from these models are provided as videos 6,7 (RTRBM) and videos 8,9 (TRBM); video 10 is a sample training sequence. Part of the Gaussian noise was removed in a manner described in [14] in both models.

## 5.3 Details of the learning procedures

Each problem was trained for 100,000 weight updates, with a momentum of 0.9, where the gradient was normalized by the length of the sequence for each gradient computation. The weights are updated after computing the gradient on a single sequence. The learning starts with $CD_{10}$ for the first 1000 weight updates, which is then switched to $CD_{25}$. The visible to hidden weights, $W$, were initialized with static $CD_5$ (without using the (R)TRBM learning rules) on 30 sequences (which resulted in 30 weight updates) with learning rate of 0.01 and momentum 0.9. These weights were then given to the (R)TRBM learning procedure, where the learning rate was linearly reduced towards 0. The weights $W'$ and the biases were initialized with a sample from spherical Gaussian of standard-deviation 0.005. For the bouncing balls problem the initial learning rate was 0.01, and for the motion capture data it was 0.005.

# 6 Conclusions

In this paper we introduced the RTRBM, which is a probabilistic model as powerful as the intractable TRBM that has an exact inference and an almost exact learning procedure. The common disadvantage of the RTRBM is that it is a recurrent neural network, a type of model known to have difficulties learning to use its hidden units to their full potential [2]. However, this disadvantage is common to many other probabilistic models, and it can be partially alleviated using techniques such as the long short term memory RNN [6].

### Acknowledgments

This research was partially supported by the Ontario Graduate Scholarship and by the Natural Council of Research and Engineering of Canada. The mocap data used in this project was obtained from `http://people.csail.mit.edu/ehsu/work/sig05stf/`. For Matlab playback of motion and generation of videos, we have adapted portions of Neil Lawrence's motion capture toolbox (`http://www.dcs.shef.ac.uk/~neil/mocap/`).

## Footnotes

[1]We use uppercase variables (as in $P(H|v)$) to denote distributions and lowercase variables (as in $P(h|v)$) to denote the (real-valued) probability $P(H = h|v)$.

[2]This is a slightly simplified description of the inference procedure in [13].

[3] When $t = 1$, $P(H_t|v_t, h_{t-1})$ stands for $P_0(H_1|v_1)$, and similarly for other conditional distributions. The same convention is used in all algorithms.

# References

[1] A.J. Bell and T.J. Sejnowski. An Information-Maximization Approach to Blind Separation and Blind Deconvolution. *Neural Computation*, 7(6):1129–1159, 1995.

[2] Y. Bengio, P. Simard, and P. Frasconi. Learning long-term dependencies with gradient descent is difficult. *Neural Networks, IEEE Transactions on*, 5(2):157–166, 1994.

[3] G.E. Hinton. Training Products of Experts by Minimizing Contrastive Divergence. *Neural Computation*, 14(8):1771–1800, 2002.

[4] G.E. Hinton, S. Osindero, and Y.W. Teh. A Fast Learning Algorithm for Deep Belief Nets. *Neural Computation*, 18(7):1527–1554, 2006.

[5] G.E. Hinton and R.R. Salakhutdinov. Reducing the Dimensionality of Data with Neural Networks. *Science*, 313(5786):504–507, 2006.

[6] S. Hochreiter and J. Schmidhuber. Long Short-Term Memory. *Neural Computation*, 9(8):1735–1780, 1997.

[7] S. Osindero and G. Hinton. Modeling image patches with a directed hierarchy of Markov random fields. *Advances Neural Information Processing Systems*, 2008.

[8] C. Peterson and J.R. Anderson. A mean field theory learning algorithm for neural networks. *Complex Systems*, 1(5):995–1019, 1987.

[9] L.R. Rabiner. A tutorial on hidden Markov models and selected applications inspeech recognition. *Proceedings of the IEEE*, 77(2):257–286, 1989.

[10] D.E. Rumelhart, G.E. Hinton, and R.J. Williams. Learning representations by back-propagating errors. *Nature*, 323(6088):533–536, 1986.

[11] R. Salakhutdinov and I. Murray. On the quantitative analysis of deep belief networks. In *Proceedings of the International Conference on Machine Learning*, volume 25, 2008.

[12] D. Sontag and T. Jaakkola. New Outer Bounds on the Marginal Polytope. *Advances in Neural Information Processing Systems*, 2008.

[13] I. Sutskever and G.E. Hinton. Learning multilevel distributed representations for high-dimensional sequences. *Proceeding of the Eleventh International Conference on Artificial Intelligence and Statistics*, pages 544–551, 2007.

[14] G.W. Taylor, G.E. Hinton, and S. Roweis. Modeling human motion using binary latent variables. *Advances in Neural Information Processing Systems*, 19:1345–1352, 2007.

[15] T. Tieleman. Training restricted boltzmann machines using approximations to the likelihood gradient. In *Proceedings of the International Conference on Machine Learning*, volume 25, 2008.

[16] M.J. Wainwright, T.S. Jaakkola, and A.S. Willsky. A new class of upper bounds on the log partition function. *IEEE Transactions on Information Theory*, 51(7):2313–2335, 2005.

[17] M.J. Wainwright and M.I. Jordan. Graphical models, exponential families, and variational inference. *UC Berkeley, Dept. of Statistics, Technical Report*, 649, 2003.

[18] M. Welling, M. Rosen-Zvi, and G. Hinton. Exponential family harmoniums with an application to information retrieval. *Advances in Neural Information Processing Systems*, 17:1481–1488, 2005.

[19] J.S. Yedidia, W.T. Freeman, and Y. Weiss. Understanding belief propagation and its generalizations. *Exploring Artificial Intelligence in the New Millennium*, pages 239–236, 2003.
